# Nonparametric Regression and Classification with Joint Sparsity Constraints

**Han Liu   John Lafferty   Larry Wasserman**

Carnegie Mellon University
Pittsburgh, PA 15213

## Abstract

We propose new families of models and algorithms for high-dimensional nonparametric learning with joint sparsity constraints. Our approach is based on a regularization method that enforces common sparsity patterns across different function components in a nonparametric additive model. The algorithms employ a coordinate descent approach that is based on a functional soft-thresholding operator. The framework yields several new models, including multi-task sparse additive models, multi-response sparse additive models, and sparse additive multi-category logistic regression. The methods are illustrated with experiments on synthetic data and gene microarray data.

## 1   Introduction

Many learning problems can be naturally formulated in terms of multi-category classification or multi-task regression. In a multi-category classification problem, it is required to discriminate between the different categories using a set of high-dimensional feature vectors—for instance, classifying the type of tumor in a cancer patient from gene expression data. In a multi-task regression problem, it is of interest to form several regression estimators for related data sets that share common types of covariates—for instance, predicting test scores across different school districts. In other areas, such as multi-channel signal processing, it is of interest to simultaneously decompose multiple signals in terms of a large common overcomplete dictionary, which is a multi-response regression problem. In each case, while the details of the estimators vary from instance to instance, across categories, or tasks, they may share a common sparsity pattern of relevant variables selected from a high-dimensional space. How to find this common sparsity pattern is an interesting learning task.

In the parametric setting, progress has been recently made on such problems using regularization based on the sum of supremum norms (Turlach et al., 2005; Tropp et al., 2006; Zhang, 2006). For example, consider the $K$-task linear regression problem $y_i^{(k)} = \beta_0^{(k)} + \sum_{j=1}^{p} \beta_j^{(k)} x_{ij}^{(k)} + \epsilon_i^{(k)}$ where the superscript $k$ indexes the tasks, and the subscript $i = 1, \ldots, n_k$ indexes the instances within a task. Using quadratic loss, Zhang (2006) suggests the following estimator

$$\widehat{\beta} = \arg\min_{\beta} \left\{ \sum_{k=1}^{K} \left[ \frac{1}{2n_k} \sum_{i=1}^{n_k} \left( y_i^{(k)} - \beta_0^{(k)} - \sum_{j=1}^{p} \beta_j^{(k)} x_{ij}^{(k)} \right)^2 \right] + \lambda \sum_{j=1}^{p} \max_{k} |\beta_j^{(k)}| \right\} \tag{1}$$

where $\max_k |\beta_j^{(k)}| = \|\beta_j\|_\infty$ is the sup-norm of the vector $\beta_j \equiv (\beta_j^{(1)}, \ldots, \beta_j^{(K)})^T$ of coefficients for the $j^{\text{th}}$ feature across different tasks. The sum of sup-norms regularization has the effect of "grouping" the elements in $\beta_j$ such that they can be shrunk towards zero simultaneously. The problems of multi-response (or multivariate) regression and multi-category classification can be viewed as a special case of the multi-task regression problem where tasks share the same design matrix. Turlach et al. (2005) and Fornasier and Rauhut (2008) propose the same sum of sup-norms

regularization as in (1) for such problems in the linear model setting. In related work, Zhang et al. (2008) propose the sup-norm support vector machine, demonstrating its effectiveness on gene data.

In this paper we develop new methods for nonparametric estimation for such multi-task and multi-category regression and classification problems. Rather than fitting a linear model, we instead estimate smooth functions of the data, and formulate a regularization framework that encourages joint functional sparsity, where the component functions can be different across tasks while sharing a common sparsity pattern. Building on a recently proposed method called sparse additive models, or "SpAM" (Ravikumar et al., 2007), we propose a convex regularization functional that can be viewed as a nonparametric analog of the sum of sup-norms regularization for linear models. Based on this regularization functional, we develop new models for nonparametric multi-task regression and classification, including multi-task sparse additive models (MT-SpAM), multi-response sparse additive models (MR-SpAM), and sparse multi-category additive logistic regression (SMALR).

The contributions of this work include (1) an efficient iterative algorithm based on a functional soft-thresholding operator derived from subdifferential calculus, leading to the multi-task and multi-response SpAM procedures, (2) a penalized local scoring algorithm that corresponds to fitting a sequence of multi-response SpAM estimates for sparse multi-category additive logistic regression, and (3) the successful application of this methodology to multi-category tumor classification and biomarker discovery from gene microarray data.

## 2 Nonparametric Models for Joint Functional Sparsity

We begin by introducing some notation. If $X$ has distribution $P_X$, and $f$ is a function of $x$, its $L_2(P_X)$ norm is denoted by $\|f\|^2 = \int_{\mathcal{X}} f^2(x) dP_X = \mathbb{E}(f^2)$. If $v = (v_1, \ldots, v_n)^T$ is a vector, define $\|v\|_n^2 = \frac{1}{n} \sum_{j=1}^n v_j^2$ and $\|v\|_\infty = \max_j |v_j|$. For a $p$-dimensional random vector $(X_1, \ldots, X_p)$, let $\mathcal{H}_j$ denote the Hilbert subspace $L_2(P_{X_j})$ of $P_{X_j}$-measurable functions $f_j(x_j)$ of the single scalar variable $X_j$ with zero mean, i.e. $\mathbb{E}[f_j(X_j)] = 0$. The inner product on this space is defined as $\langle f_j, g_j \rangle = \mathbb{E}[f_j(X_j) g_j(X_j)]$. In this paper, we mainly study multivariate functions $f(x_1, \ldots, x_p)$ that have an additive form, i.e., $f(x_1, \ldots, x_p) = \alpha + \sum_j f_j(x_j)$, with $f_j \in \mathcal{H}_j$ for $j = 1, \ldots, p$. With $\mathcal{H} \equiv \{1\} \oplus \mathcal{H}_1 \oplus \mathcal{H}_2 \oplus \ldots \oplus \mathcal{H}_p$ denoting the direct sum Hilbert space, we have that $f \in \mathcal{H}$.

### 2.1 Multi-task/Multi-response Sparse Additive Models

In a $K$-task regression problem, we have observations $\{(x_i^{(k)}, y_i^{(k)}), i = 1, \ldots, n_k, k = 1, \ldots, K\}$, where $x_i^{(k)} = (x_{i1}^{(k)}, \ldots, x_{ip}^{(k)})^T$ is a $p$-dimensional covariate vector, the superscript $k$ indexes tasks and $i$ indexes the i.i.d. samples for each task. In the following, for notational simplicity, we assume that $n_1 = \ldots = n_K = n$. We also assume different tasks are comparable and each $Y^{(k)}$ and $X_j^{(k)}$ has been standardized, i.e., has mean zero and variance one. This is not really a restriction of the model since a straightforward weighting scheme can be adopted to extend our approach to handle noncomparable tasks. We assume the true model is $\mathbb{E}\left(Y^{(k)} \mid X^{(k)} = x^{(k)}\right) = f^{(k)}(x^{(k)}) \equiv \sum_{j=1}^p f_j^{(k)}(x_j^{(k)})$ for $k = 1, \ldots, K$, where, for simplicity, we take all intercepts $\alpha^{(k)}$ to be zero. Let $\mathcal{Q}_{f^{(k)}}(x, y) = (y - f^{(k)}(x))^2$ denote the quadratic loss. To encourage common sparsity patterns across different function components, we define the regularization functional $\Phi_K(f)$ by

$$\Phi_K(f) = \sum_{j=1}^p \max_{k=1,\ldots,K} \|f_j^{(k)}\|. \tag{2}$$

The regularization functional $\Phi_K(f)$ naturally combines the idea of the sum of sup-norms penalty for parametric joint sparsity and the regularization idea of SpAM for nonparametric functional sparsity; if $K = 1$, then $\Phi_1(f)$ is just the regularization term introduced for (single-task) sparse additive models by Ravikumar et al. (2007). If each $f_j^{(k)}$ is a linear function, then $\Phi_K(f)$ reduces to the sum of sup-norms regularization term as in (1). We shall employ $\Phi_K(f)$ to induce joint functional sparsity in nonparametric multi-task inference.

Using this regularization functional, the multi-task sparse additive model (MT-SpAM) is formulated as a penalized M-estimator, by framing the following optimization problem

$$\widehat{f}^{(1)}, \ldots, \widehat{f}^{(K)} = \underset{f^{(1)}, \ldots, f^{(K)}}{\arg\min} \left\{ \frac{1}{2n} \sum_{i=1}^{n} \sum_{k=1}^{K} \mathcal{Q}_{f^{(k)}}(x_i^{(k)}, y_i^{(k)}) + \lambda \Phi_K(f) \right\} \quad (3)$$

where $f_j^{(k)} \in \mathcal{H}_j^{(k)}$ for $j = 1, \ldots, p$ and $k = 1, \ldots, K$, and $\lambda > 0$ is a regularization parameter. The multi-response sparse additive model (MR-SpAM) has exactly the same formulation as in (3) except that a common design matrix is used across the $K$ different tasks.

## 2.2 Sparse Multi-Category Additive Logistic Regression

In a $K$-category classification problem, we are given $n$ examples $(x_1, y_1), \ldots, (x_n, y_n)$ where $x_i = (x_{i1}, \ldots, x_{ip})^T$ is a $p$-dimensional predictor vector and $y_i = (y_i^{(1)}, \ldots, y_i^{(K-1)})^T$ is a $(K-1)$-dimensional response vector in which at most one element can be one, with all the others being zero. Here, we adopt the common "1-of-$K$" labeling convention where $y_i^{(k)} = 1$ if $x_i$ has category $k$ and $y_i^{(k)} = 0$ otherwise; if all elements of $y_i$ are zero, then $x_i$ is assigned the $K$-th category.

The multi-category additive logistic regression model is

$$\mathbb{P}(Y^{(k)} = 1 \mid X = x) = \frac{\exp\left(f^{(k)}(x)\right)}{1 + \sum_{k'=1}^{K-1} \exp\left(f^{(k')}(x)\right)}, \quad k = 1, \ldots, K-1 \quad (4)$$

where $f^{(k)}(x) = \alpha^{(k)} + \sum_{j=1}^{p} f_j^{(k)}(x_j)$ has an additive form. We define $f = (f^{(1)}, \ldots, f^{(K-1)})$ to be a discriminant function and $p_f^{(k)}(x) = \mathbb{P}(Y^{(k)} = 1 \mid X = x)$ to be the conditional probability of category $k$ given $X = x$. The logistic regression classifier $h_f(\cdot)$ induced by $f$, which is a mapping from the sample space to the category labels, is simply given by $h_f(x) = \arg\max_{k=1,\ldots,K} p_f^{(k)}(x)$. If a variable $X_j$ is irrelevant, then all of the component functions $f_j^{(k)}$ are identically zero, for each $k = 1, 2, \ldots, K-1$. This motivates the use of the regularization functional $\Phi_{K-1}(f)$ to zero out entire vectors $f_j = (f_j^{(1)}, \ldots, f_j^{(K-1)})$.

Denoting

$$\ell_f(x, y) = \sum_{k=1}^{K-1} y^{(k)} f^{(k)}(x) - \log\left(1 + \sum_{k'=1}^{K-1} \exp f^{(k')}(x)\right)$$

as the multinomial log-loss, the sparse multi-category additive logistic regression estimator (SMALR) is thus formulated as the solution to the optimization problem

$$\widehat{f}^{(1)}, \ldots, \widehat{f}^{(K-1)} = \underset{f^{(1)}, \ldots, f^{(K-1)}}{\arg\min} \left\{ -\frac{1}{n} \sum_{i=1}^{n} \ell_f(x_i, y_i) + \lambda \Phi_{K-1}(f) \right\} \quad (5)$$

where $f_j^{(k)} \in \mathcal{H}_j^{(k)}$ for $j = 1, \ldots, p$ and $k = 1, \ldots, K-1$.

# 3  Simultaneous Sparse Backfitting

We use a blockwise coordinate descent algorithm to minimize the functional defined in (3). We first formulate the population version of the problem by replacing sample averages by their expectations. We then derive stationary conditions for the optimum and obtain a population version algorithm for computing the solution by a series of soft-thresholded univariate conditional expectations. Finally, a finite sample version of the algorithm can be derived by plugging in nonparametric smoothers for these conditional expectations.

For the $j^{\text{th}}$ block of component functions $f_j^{(1)}, \ldots, f_j^{(K)}$, let $R_j^{(k)} = Y^{(k)} - \sum_{l \neq j} f_l^{(k)}(X_l^{(k)})$ denote the partial residuals. Assuming all but the functions in the $j^{\text{th}}$ block to be fixed, the optimization problem is reduced to

$$\widehat{f}_j^{(1)}, \ldots, \widehat{f}_j^{(K)} = \underset{f_j^{(1)}, \ldots, f_j^{(K)}}{\arg\min} \left\{ \frac{1}{2} \mathbb{E}\left[ \sum_{k=1}^{K} \left( R_j^{(k)} - f_j^{(k)}(X_j^{(k)}) \right)^2 \right] + \lambda \max_{k=1,\ldots,K} \|f_j^{(k)}\| \right\}. \quad (6)$$

The following result characterizes the solution to (6).

**Theorem 1.** *Let $P_j^{(k)} = \mathbb{E}\left(R_j^{(k)} \mid X_j^{(k)}\right)$ and $s_j^{(k)} = \|P_j^{(k)}\|$, and order the indices according to $s_j^{(k_1)} \geq s_j^{(k_2)} \geq \ldots \geq s_j^{(k_K)}$. Then the solution to (6) is given by*

$$
f_j^{(k_i)} = \begin{cases} P_j^{(k_i)} & \text{for } i > m^* \\ \dfrac{1}{m^*}\left[\displaystyle\sum_{i'=1}^{m^*} s_j^{(k_{i'})} - \lambda\right]_+ \dfrac{P_j^{(k_i)}}{s_j^{(k_i)}} & \text{for } i \leq m^*. \end{cases} \tag{7}
$$

*where $m^* = \arg\max_m \frac{1}{m}\left(\sum_{i'=1}^m s_j^{(k_{i'})} - \lambda\right)$ and $[\cdot]_+$ denotes the positive part.*

Therefore, the optimization problem in (6) is solved by a soft-thresholding operator, given in equation (7), which we shall denote as

$$
(f_j^{(1)}, \ldots, f_j^{(K)}) = \text{Soft}_\lambda^{(\infty)}[R_j^{(1)}, \ldots, R_j^{(K)}]. \tag{8}
$$

While the proof of this result is lengthy, we sketch the key steps below, which are a functional extension of the subdifferential calculus approach of Fornasier and Rauhut (2008) in the linear setting. First, we formulate an optimality condition in terms of the Gâteaux derivative as follows.

**Lemma 2.** *The functions $f_j^{(k)}$ are solutions to (6) if and only if $f_j^{(k)} - P_j^{(k)} + \lambda u_k v_k = 0$ (almost surely), for $k = 1, \ldots, K$, where $u_k$ are scalars and $v_k$ are measurable functions of $X_j^{(k)}$, with*

$$
(u_1, \ldots, u_K)^T \in \partial\|\cdot\|_\infty\Big|_{\left(\|f_j^{(1)}\|, \ldots, \|f_j^{(K)}\|\right)^T} \text{ and } v_k \in \partial\|f_j^{(k)}\|, \ k = 1, \ldots, K.
$$

Here the former one denotes the subdifferential of the convex functional $\|\cdot\|_\infty$ evaluated at $(\|f_j^{(1)}\|, \ldots, \|f_j^{(K)}\|)^T$, it lies in a $K$-dimensional Euclidean space. And the latter denotes the subdifferential of $\|f_j^{(k)}\|$, which is a set of functions. Next, the following proposition from Rockafellar and Wets (1998) is used to characterize the subdifferential of sup-norms.

**Lemma 3.** *The subdifferential of $\|\cdot\|_\infty$ on $\mathbb{R}^K$ is*

$$
\partial\|\cdot\|_\infty\big|_x = \begin{cases} B^1(1) & \text{if } x = \mathbf{0} \\ \text{conv}\{\text{sign}(x_k)e_k : |x_k| = \|x\|_\infty\} & \text{otherwise.} \end{cases}
$$

*where $B^1(1)$ denotes the $\ell_1$ ball of radius one, $\text{conv}(A)$ denotes the convex hull of set $A$, and $e_k$ is the $k$-th canonical unit vector in $\mathbb{R}^K$.*

Using Lemma 2 and Lemma 3, the proof of Theorem 1 proceeds by considering three cases for the sup-norm subdifferential evaluated at $(\|f_j^{(1)}\|, \ldots, \|f_j^{(K)}\|)^T$: (1) $\|f_j^{(k)}\| = 0$ for $k = 1, \ldots, K$; (2) there exists a unique $k$, such that $\|f_j^{(k)}\| = \max_{k'=1,\ldots,K} \|f_j^{(k')}\| \neq 0$; (3) there exists at least two $k \neq k'$, such that $\|f_j^{(k)}\| = \|f_j^{(k')}\| = \max_{m=1,\ldots,K} \|f_j^{(m)}\| \neq 0$. The derivations for cases (1) and (2) are relatively straightforward, but for case (3) we prove the following.

**Lemma 4.** *The sup-norm is attained precisely at $m > 1$ entries if only if $m$ is the largest number such that $s_j^{(k_m)} \geq \frac{1}{m-1}\left(\sum_{i'=1}^{m-1} s_j^{(k_{i'})} - \lambda\right)$.*

The proof of Theorem 1 then follows from the above lemmas and some calculus. Based on this result, the data version of the soft-thresholding operator is obtained by replacing the conditional expectation $P_j^{(k)} = \mathbb{E}(R_j^{(k)} \mid X_j^{(k)})$ by $\mathcal{S}_j^{(k)} R_j^{(k)}$, where $\mathcal{S}_j^{(k)}$ is a nonparametric smoother for variable $X_j^{(k)}$, e.g., a local linear or spline smoother; see Figure 1. The resulting simultaneous sparse backfitting algorithm for multi-task and multi-response sparse additive models (MT-SpAM and MR-SpAM) is shown in Figure 2. The algorithm for the multi-response case (MR-SpAM) has $\mathcal{S}_j^{(1)} = \ldots = \mathcal{S}_j^{(K)}$ since there is only a common design matrix.

SOFT-THRESHOLDING OPERATOR $\text{SOFT}_\lambda^{(\infty)}[R_j^{(1)},\ldots,R_j^{(K)};\mathcal{S}_j^{(1)},\ldots,\mathcal{S}_j^{(K)}]$: DATA VERSION

---

*Input*: Smoothing matrices $\mathcal{S}_j^{(k)}$, residuals $R_j^{(k)}$ for $k = 1,\ldots,K$, regularization parameter $\lambda$.

(1) Estimate $P_j^{(k)} = \mathbb{E}\left[R_j^{(k)} \mid X_j^{(k)}\right]$ by smoothing: $\widehat{P}_j^{(k)} = \mathcal{S}_j^{(k)} R_j^{(k)}$;

(2) Estimate norm: $\widehat{s}_j^{(k)} = \|\widehat{P}_j\|_n$ and order the indices according to $\widehat{s}_j^{(k_1)} \geq \widehat{s}_j^{(k_2)} \geq \ldots \geq \widehat{s}_j^{(k_K)}$;

(3) Find $m^* = \arg\max_m \frac{1}{m}\left(\sum_{i'=1}^{m} s_j^{(k_{i'})} - \lambda\right)$ and calculate

$$\widehat{f}_j^{(k_i)} = \begin{cases} \widehat{P}_j^{(k_i)} & \text{for } i > m^* \\ \dfrac{1}{m^*}\left[\displaystyle\sum_{i'=1}^{m^*}\widehat{s}_j^{(k_{i'})} - \lambda\right]_+ \dfrac{\widehat{P}_j^{(k_i)}}{\widehat{s}_j^{(k_i)}} & \text{for } i \leq m^*; \end{cases}$$

(4) Center $\widehat{f}_j^{(k)} \leftarrow \widehat{f}_j^{(k)} - \text{mean}(\widehat{f}_j^{(k)})$ for $k = 1,\ldots,K$.

*Output*: Functions $\widehat{f}_j^{(k)}$ for $k = 1,\ldots,K$.

---

Figure 1: Data version of the soft-thresholding operator.

MULTI-TASK AND MULTI-RESPONSE SPAM

---

*Input*: Data $(x_i^{(k)}, y_i^{(k)})$, $i = 1,\ldots,n$, $k = 1,\ldots,K$ and regularization parameter $\lambda$.

*Initialize*: Set $\widehat{f}_j^{(k)} = 0$ and compute smoothers $\mathcal{S}_j^{(k)}$ for $j = 1,\ldots,p$ and $k = 1,\ldots,K$;

*Iterate* until convergence:

    *For each $j = 1,\ldots,p$*:

(1) Compute residuals: $R_j^{(k)} = y^{(k)} - \sum_{k' \neq j} \widehat{f}_{k'}^{(k)}$ for $k = 1,\ldots,K$;

(2) Threshold: $\widehat{f}_j^{(1)},\ldots,\widehat{f}_j^{(K)} \leftarrow \text{Soft}_\lambda^{(\infty)}[R_j^{(1)},\ldots,R_j^{(K)};\mathcal{S}_j^{(1)},\ldots,\mathcal{S}_j^{(K)}]$.

*Output*: Functions $\widehat{f}^{(k)}$ for $k = 1,\ldots,K$.

---

Figure 2: The simultaneous sparse backfitting algorithm for MT-SpAM or MR-SpAM. For the multi-response case, the same smoothing matrices are used for each $k$.

## 3.1 Penalized Local Scoring Algorithm for SMALR

We now derive a penalized local scoring algorithm for sparse multi-category additive logistic regression (SMALR), which can be viewed as a variant of Newton's method in function space. At each iteration, a quadratic approximation to the loss is used as a surrogate functional with the regularization term added to induce joint functional sparsity. However, a technical difficulty is that the approximate quadratic problem in each iteration is weighted by a non-diagonal matrix in function space, thus a trivial extension of the algorithm in (Ravikumar et al., 2007) for sparse binary nonparametric logistic regression does not apply. To tackle this problem, we use an auxiliary function to lower bound the log-loss, as in (Krishnapuram et al., 2005).

The population version of the log-loss is $L(f) = \mathbb{E}[\ell_f(X,Y)]$ with $f = (f^{(1)},\ldots,f^{(K-1)})$. A second-order Lagrange form Taylor expansion to $L(f)$ at $\widehat{f}$ is then

$$L(f) = L(\widehat{f}) + \mathbb{E}\left[\nabla L(\widehat{f})^T(f - \widehat{f})\right] + \frac{1}{2}\mathbb{E}\left[(f - \widehat{f})^T H(\widetilde{f})(f - \widehat{f})\right] \tag{9}$$

for some function $\widetilde{f}$, where the gradient is $\nabla L(\widehat{f}) = Y - p_{\widehat{f}}(X)$ with $p_{\widehat{f}}(X) = (p_{\widehat{f}}(Y^{(1)} = 1 \mid X),\ldots,p_{\widehat{f}}(Y^{(K-1)} = 1 \mid X))^T$, and the Hessian is $H(\widetilde{f}) = -\text{diag}\left(p_{\widetilde{f}}(X)\right) + p_{\widetilde{f}}(X)p_{\widetilde{f}}(X)^T$. Defining $B = -(1/4)I_{K-1}$, it is straightforward to show that $B \preceq H(\widetilde{f})$, i.e., $H(\widetilde{f}) - B$ is positive-definite. Therefore, we have that

$$L(f) \geq L(\widehat{f}) + \mathbb{E}\left[\nabla L(\widehat{f})^T(f - \widehat{f})\right] + \frac{1}{2}\mathbb{E}\left[(f - \widehat{f})^T B(f - \widehat{f})\right]. \tag{10}$$

*Input*: Data $(x_i, y_i), i = 1, \ldots, n$ and regularization parameter $\lambda$.

*Initialize*: $\widehat{f}_j^{(k)} = 0$ and $\widehat{\alpha}^{(k)} = \log\left(\sum_{i=1}^n y_i^{(k)} \Big/ \left(n - \sum_{i=1}^n \sum_{k'=1}^{K-1} y_i^{(k')}\right)\right), k = 1, \ldots, K - 1$

*Iterate* until convergence:

(1) Compute $p_{\widehat{f}}^{(k)}(x_i) \equiv \mathbb{P}(Y^{(k)} = 1 \mid X = x_i)$ as in (4) for $k = 1, \ldots, K - 1$;

(2) Calculate the transformed responses $Z_i^{(k)} = 4\left(y_i^{(k)} - p_{\widehat{f}}^{(k)}(x_i)\right) + \widehat{\alpha}^{(k)} + \sum_{j=1}^p \widehat{f}_j^{(k)}(x_{ij})$
for $k = 1, \ldots, K - 1$ and $i = 1, \ldots, n$;

(3) Call subroutines $(\widehat{f}^{(1)}, \ldots, \widehat{f}^{(K-1)}) \leftarrow \text{MR-SpAM}\left((x_i, Z_i^{(k)})_{i=1}^n, \sqrt{2}\lambda\right)$;

(4) Adjust the intercepts: $\alpha^{(k)} \leftarrow \dfrac{1}{n}\sum_{i=1}^n Z_i^{(k)}$;

*Output*: Functions $\widehat{f}^{(k)}$ and intercepts $\widehat{\alpha}^{(k)}$ for $k = 1, \ldots, K - 1$.

Figure 3: The penalized local scoring algorithm for SMALR.

The following lemma results from straightforward calculation.

**Lemma 5.** *The solution $f$ that maximizes the righthand side of (10) is equivalent to the solution that minimizes $\frac{1}{2}\mathbb{E}\left(\|Z - Af\|_n^2\right)$ where $A = (-B)^{1/2}$ and $Z = A^{-1}(Y - p_{\widehat{f}}) + A\widehat{f}$.*

Recalling that $f^{(k)} = \alpha^{(k)} + \sum_{j=1}^p f_j^{(k)}$, equation (9) and Lemma 5 then justify the use of the auxiliary functional

$$\frac{1}{2}\sum_{k=1}^{K-1}\mathbb{E}\left[\left(Z'^{(k)} - \sum_{j=1}^p f^{(k)}(X_j)\right)^2\right] + \lambda'\Phi_{K-1}(f) \tag{11}$$

where $Z'^{(k)} = 4\left(Y^{(k)} - \mathbb{P}_{\widehat{f}}(Y^{(k)} = 1 \mid X)\right) + \widehat{\alpha}^{(k)} + \sum_{j=1}^p \widehat{f}_j^{(k)}(X_j)$ and $\lambda' = \sqrt{2}\lambda$. This is precisely in the form of a multi-response SpAM optimization problem in equation (3). The resulting algorithm, in the finite sample case, is shown in Figure 3.

## 4 Experiments

In this section, we first use simulated data to investigate the performance of the MT-SpAM simultaneous sparse backfitting algorithm. We then apply SMALR to a tumor classification and biomarker identification problem. In all experiments, the data are rescaled to lie in the $p$-dimensional cube $[0,1]^p$. We use local linear smoothing with a Gaussian kernel. To choose the regularization parameter $\lambda$, we simply use $J$-fold cross-validation or the GCV score from (Ravikumar et al., 2007) extended to the multi-task setting: $\text{GCV}(\lambda) = \sum_{i=1}^n \sum_{k=1}^K \mathcal{Q}_{\widehat{f}^{(k)}}(x_i^{(k)}, y_i^{(k)}))/(n^2 K^2 - (nK)\text{df}(\lambda))^2$ where $\text{df}(\lambda) = \sum_{k=1}^K \sum_{j=1}^p \nu_j^{(k)} I\left(\|\widehat{f}_j^{(k)}\|_n \neq 0\right)$, and $\nu_j^{(k)} = \text{trace}(\mathcal{S}_j^{(k)})$ is the effective degrees of freedom for the univariate local linear smoother on the $j^{\text{th}}$ variable.

### 4.1 Synthetic Data

We generated $n = 100$ observations from a 10-dimensional three-task additive model with four relevant variables: $y_i^{(k)} = \sum_{j=1}^4 f_j^{(k)}(x_{ij}^{(k)}) + \epsilon_i^{(k)}, k = 1, 2, 3$, where $\epsilon_i^{(k)} \sim \mathcal{N}(0, 1)$; the component functions $f_j^{(k)}$ are plotted in Figure 4. The 10-dimensional covariates are generated as $X_j^{(k)} = (W_j^{(k)} + tU^{(k)})/(1 + t), j = 1, \ldots, 10$ where $W_1^{(k)}, \ldots, W_{10}^{(k)}$ and $U^{(k)}$ are i.i.d. sampled from Uniform$(-2.5, 2.5)$. Thus, the correlation between $X_j$ and $X_{j'}$ is $t^2/(1 + t^2)$ for $j \neq j'$.

The results of applying MT-SpAM with the bandwidths $h = (0.08, \ldots, 0.08)$ and regularization parameter $\lambda = 0.25$ are summarized in Figure 4. The upper 12 figures show the 12 relevant component functions for the three tasks; the estimated function components are plotted as solid black

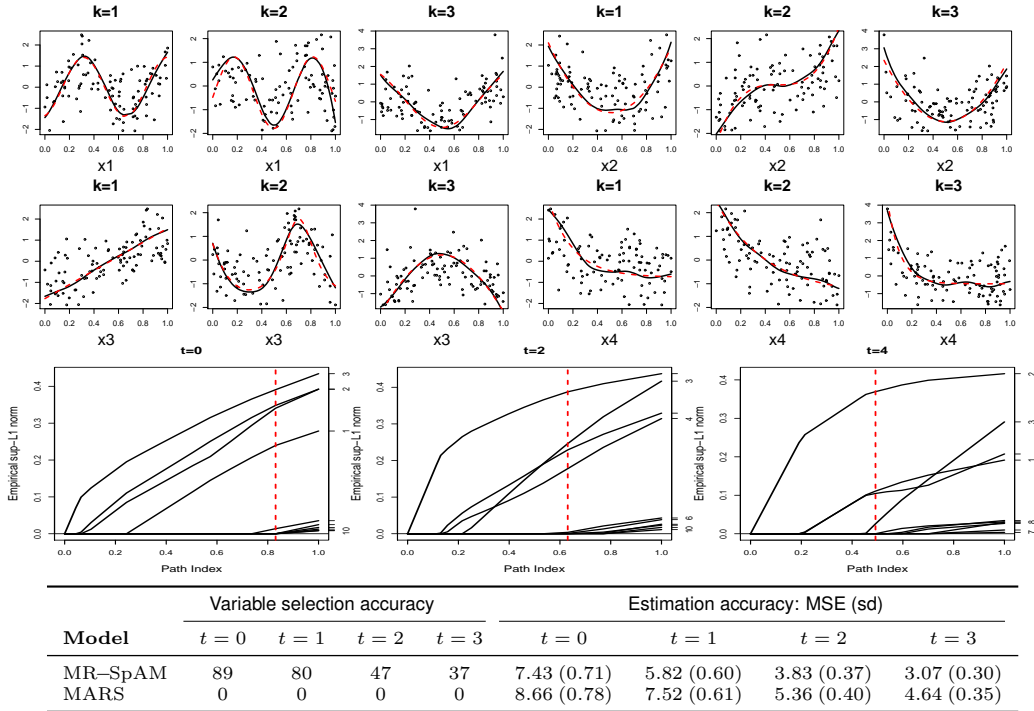

| | Variable selection accuracy | | | | Estimation accuracy: MSE (sd) | | | |
|---|---|---|---|---|---|---|---|---|
| **Model** | $t=0$ | $t=1$ | $t=2$ | $t=3$ | $t=0$ | $t=1$ | $t=2$ | $t=3$ |
| MR–SpAM | 89 | 80 | 47 | 37 | 7.43 (0.71) | 5.82 (0.60) | 3.83 (0.37) | 3.07 (0.30) |
| MARS | 0 | 0 | 0 | 0 | 8.66 (0.78) | 7.52 (0.61) | 5.36 (0.40) | 4.64 (0.35) |

Figure 4: (Top) Estimated vs. true functions from MT-SpAM; (Middle) Regularization paths using MT-SpAM. (Bottom) Quantitative comparison between MR-SpAM and MARS

lines and the true function components are plotted using dashed red lines. For all the other variables (from dimension 5 to dimension 10), both the true and estimated components are zero. The middle three figures show regularization paths as the parameter $\lambda$ varies; each curve is a plot of the maximum empirical $L_1$ norm of the component functions for each variable, with the red vertical line representing the selected model using the GCV score. As the correlation increases ($t$ increases), the separation between the relevant dimensions and the irrelevant dimensions becomes smaller. Using the same setup but with one common design matrix, we also compare the quantitative performance of MR-SpAM with MARS (Friedman, 1991), which is a popular method for multi-response additive regression. Using 100 simulations, the table illustrates the number of times the models are correctly identified and the mean squared error with the standard deviation in the parentheses. (The MARS simulations are carried out in R, using the default options of the `mars` function in the `mda` library.)

## 4.2 Gene Microarray Data

Here we apply the sparse multi-category additive logistic regression model to a microarray dataset for small round blue cell tumors (SRBCT) (Khan et al., 2001). The data consist of expression profiles of 2,308 genes (Khan et al., 2001) with tumors classified into 4 categories: neuroblastoma (NB), rhabdomyosarcoma (RMS), non-Hodgkin lymphoma (NHL), and the Ewing family of tumors (EWS). The dataset includes a training set of size 63 and a test set of size 20. These data have been analyzed by different groups. The main purpose is to identify important biomarkers, which are a small set of genes that can accurately predict the type of tumor of a patient. To achieve 100% accuracy on the test data, Khan et al. (2001) use an artificial neural network approach to identify 96 genes. Tibshirani et al. (2002) identify a set of only 43 genes, using a method called nearest shrunken centroids. Zhang et al. (2008) identify 53 genes using the sup-norm support vector machine.

In our experiment, SMALR achieves 100% prediction accuracy on the test data with only 20 genes, which is a much smaller set of predictors than identified in the previous approaches. We follow the same procedure as in (Zhang et al., 2008), and use a very simple screening step based on the marginal correlation to first reduce the number of genes to 500. The SMALR model is then trained using a plugin bandwidth $h_0 = 0.08$, and the regularization parameter $\lambda$ is tuned using 4-fold cross validation. The results are tabulated in Figure 5. In the left figure, we show a "heat map" of the selected variables on the training set. The rows represent the selected genes, with their cDNA chip image id. The patients are grouped into four categories according to the corresponding tumors,

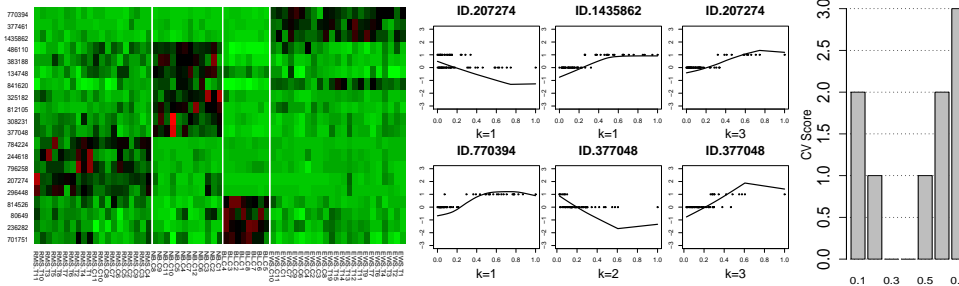

Figure 5: SMALR results on gene data: heat map (left), marginal fits (center), and CV score (right).

as illustrated in the vertical groupings. The genes are ordered by hierarchical clustering of their expression profiles. The heatmap clearly shows four block structures for the four tumor categories. This suggests visually that the 20 genes selected are highly informative of the tumor type. In the middle of Figure 5, we plot the fitted discriminant functions of different genes, with their image ids listed on the plot. The values $k = 1, 2, 3$ under each subfigure indicate the discriminant function the plot represents. We see that the fitted functions are nonlinear. The right subfigure illustrates the total number of misclassified samples using 4-fold cross validation, the $\lambda$ values $0.3, 0.4$ are both zero, for the purpose of a sparser biomarker set, we choose $\lambda = 0.4$. Interestingly, only 10 of the 20 identified genes from our method are among the 43 genes selected using the shrunken centroids approach of Tibshirani et al. (2002). 16 of them are are among the 96 genes selected by neural network approach of Khan et al. (2001). This non-overlap may suggest some further investigation.

## 5   Discussion and Acknowledgements

We have presented new approaches to fitting sparse nonparametric multi-task regression models and sparse multi-category classification models. Due to space constraints, we have not discussed results on the statistical properties of these methods, such as oracle inequalities and risk consistency; these theoretical results will be reported elsewhere. This research was supported in part by NSF grant CCF-0625879.

## References

FORNASIER, M. and RAUHUT, H. (2008). Recovery algorithms for vector valued data with joint sparsity constraints. *SIAM Journal of Numerical Analysis* **46** 577–613.

FRIEDMAN, J. H. (1991). Multivariate adaptive regression splines. *The Annals of Statistics* **19** 1–67.

KHAN, J., WEI, J. S., RINGNER, M., SAA, L. H., LADANYI, M., WESTERMANN, F., BERTHOLD, F., SCHWAB, M., ANTONESCU, C. R., PETERSON, C. and MELTZER, P. S. (2001). Classification and diagnostic prediction of cancers using gene expression profiling and artificial neural networks. *Nature Medicine* **7** 673 –679.

KRISHNAPURAM, B., CARIN, L., FIGUEIREDO, M. and HARTEMINK, A. (2005). Sparse multinomial logistic regression: Fast algorithms and generalization bounds. *IEEE Transactions on Pattern Analysis and Machine Intelligence* **27** 957– 968.

RAVIKUMAR, P., LIU, H., LAFFERTY, J. and WASSERMAN, L. (2007). SpAM: Sparse additive models. In *Advances in Neural Information Processing Systems 20*. MIT Press.

ROCKAFELLAR, R. T. and WETS, R. J.-B. (1998). *Variational Analysis*. Springer-Verlag Inc.

TIBSHIRANI, R., HASTIE, T., NARASIMHAN, B., and CHU, G. (2002). Diagnosis of multiple cancer types by shrunken centroids of gene expression. *Proc Natl Acad Sci U.S.A.* **99** 6567–6572.

TROPP, J., GILBERT, A. C. and STRAUSS, M. J. (2006). Algorithms for simultaneous sparse approximation. Part II: Convex relaxation. *Signal Processing* **86** 572–588.

TURLACH, B., VENABLES, W. N. and WRIGHT, S. J. (2005). Simultaneous variable selection. *Technometrics* **27** 349–363.

ZHANG, H. H., LIU, Y., WU, Y. and ZHU, J. (2008). Variable selection for the multicategory SVM via adaptive sup-norm regularization. *Electronic Journal of Statistics* **2** 149–1167.

ZHANG, J. (2006). A probabilistic framework for multitask learning. Tech. Rep. CMU-LTI-06-006, Ph.D. thesis, Carnegie Mellon University.

